# Non-Stochastic Bandit Slate Problems

**Satyen Kale**
Yahoo! Research
Santa Clara, CA
skale@yahoo-inc.com

**Lev Reyzin**[*]
Georgia Inst. of Technology
Atlanta, GA
lreyzin@cc.gatech.edu

**Robert E. Schapire**[†]
Princeton University
Princeton, NJ
schapire@cs.princeton.edu

## Abstract

We consider bandit problems, motivated by applications in online advertising and news story selection, in which the learner must repeatedly select a *slate*, that is, a subset of size $s$ from $K$ possible actions, and then receives rewards for just the selected actions. The goal is to minimize the regret with respect to total reward of the best slate computed in hindsight. We consider unordered and ordered versions of the problem, and give efficient algorithms which have regret $O(\sqrt{T})$, where the constant depends on the specific nature of the problem. We also consider versions of the problem where we have access to a number of policies which make recommendations for slates in every round, and give algorithms with $O(\sqrt{T})$ regret for competing with the best such policy as well. We make use of the technique of relative entropy projections combined with the usual multiplicative weight update algorithm to obtain our algorithms.

## 1 Introduction

In traditional bandit models, the learner is presented with a set of $K$ actions. On each of $T$ rounds, an adversary (or the world) first chooses rewards for each action, and afterwards the learner decides which action it wants to take. The learner then receives the reward of its chosen action, but does not see the rewards of the other actions. In the standard bandit setting, the learner's goal is to compete with the best fixed arm in hindsight. In the more general "experts setting," each of $N$ experts recommends an arm on each round, and the goal of the learner is to perform as well as the best expert in hindsight.

The bandit setting tackles many problems where a learner's decisions reflect not only how well it performs but also the data it learns from — a good algorithm will balance exploiting actions it already knows to be good and exploring actions for which its estimates are less certain. One such real-world problem appears in computational advertising, where publishers try to present their customers with relevant advertisements. In this setting, the actions correspond to advertisements, and choosing an action means displaying the corresponding ad. The rewards correspond to the payments from the advertiser to the publisher, and these rewards depend on the probability of users clicking on the ads.

Unfortunately, many real-world problems, including the computational advertising problem, do not fit so nicely into the traditional bandit framework. Most of the time, advertisers have the ability to display more than one ad to users, and users can click on more than one of the ads displayed to them. To capture this reality, in this paper we define the **slate problem**. This setting is similar to the traditional bandit setting, except that here the advertiser selects a slate, or subset, of $S$ actions.

In this paper we first consider the **unordered** slate problem, where the reward to the learning algorithm is the sum of the rewards of the chosen actions in the slate. This setting is applicable when all

---

[*]This work was done while Lev Reyzin was at Yahoo! Research, New York. This material is based upon work supported by the National Science Foundation under Grant #0937060 to the Computing Research Association for the Computing Innovation Fellowship program.

[†]This work was done while R. Schapire was visiting Yahoo! Research, New York.

actions in a slate are treated equally. While this is a realistic assumption in certain settings, we also deal with the case when different positions in a slate have different importance. Going back to our computational advertising example, we can see not all ads are given the same treatment (i.e. an ad displayed higher in a list is more likely to be clicked on). One may plausibly assume that for every ad and every position that it can be shown in, there is a click-through-rate associated with the (ad, position) pair, which specifies the probability that a user will click on the ad if it is displayed in that position. This is a very general user model used widely in practice in web search engines. To abstract this, we turn to the **ordered** slate problem, where for each action and position in the ordering, the adversary specifies a reward for using the action in that position. The reward to the learner then is the sum of the rewards of the (actions, position) pairs in the chosen ordered slate.[1] This setting is similar to that of György, Linder, Lugosi and Ottucsák [10] in that the cost of *all actions* in the chosen slate are revealed, rather than just the total cost of the slate.

Finally, we show how to tackle these problems in the experts setting, where instead of competing with the best slate in hindsight, the algorithm competes with the best expert, recommending different slates on different rounds.

One key idea appearing in our algorithms is to use a variant of the multiplicative weights expert algorithm for a restricted convex set of distributions. In our case, the restricted set of distributions over actions corresponds to the one defined by the stipulation that the learner choose a slate instead of individual actions. Our variant first finds the distribution generated by multiplicative weights and then chooses the closest distribution in the restricted subset using relative entropy as the distance metric — this is a type of Bregman projection, which has certain nice properties for our analysis.

**Previous Work.** The multi-armed bandit problem, first studied by Lai and Robbins [15], is a classic problem which has had wide application. In the stochastic setting, where the rewards of the arms are i.i.d., Lai and Robbins [15] and Auer, Cesa-Bianchi and Fischer [2] gave regret bounds of $O(K \ln(T))$. In the non-stochastic setting, Auer et al. [3] gave regret bounds of $O(\sqrt{K \ln(K)T})$.[2] This non-stochastic setting of the multi-armed bandit problem is exactly the specific case of our problem when the slate size is $1$, and hence our results generalize those of Auer et al. which can be recovered by setting $s = 1$.

Our problem is a special case of the more general online linear optimization with bandit feedback problem [1, 4, 5, 11]. Specializing the best result in this series to our setting, we get worse regret bounds of $O(\sqrt{T \log(T)})$. The constant in the $O(\cdot)$ notation is also worse than our bounds. For a more specific comparison of regret bounds, see Section 2. Our algorithms, being specialized for the slates problem, are simpler to implement as well, avoiding the sophisticated self-concordant barrier techniques of [1].

This work also builds upon the algorithm in [18] to learn subsets of experts and the algorithm in [12] for learning permutations, both in the full information setting. Our work is also a special case of the Combinatorial Bandits setting of Cesa-Bianchi and Lugosi [9]; however, our algorithms obtain better regret bounds and are computationally more efficient.

Our multiplicative weights algorithm also appears under the name Component Hedge in the independent work of Koolen, Warmuth and Kivinen [14]. Furthermore, the expertless, unordered slate problem is studied by Uchiya, Nakamura and Kudo [17] who obtain the same asymptotic bounds as appear in this paper, though using different techniques.

## 2    Statement of the problem and main results

**Notation.** For vectors $\mathbf{x}, \mathbf{y} \in \mathbb{R}^K$, $\mathbf{x} \cdot \mathbf{y}$ denotes their inner product, viz. $\sum_i x_i y_i$. For matrices $\mathbf{X}, \mathbf{Y} \in \mathbb{R}^{s \times K}$, $\mathbf{X} \bullet \mathbf{Y}$ denotes their inner product considering them vectors in $\mathbb{R}^{sK}$, viz.

$\sum_{ij} X_{ij} Y_{ij}$. For a set $S$ of actions, let $\mathbf{1}_S$ be the indicator vector for that set. For two distributions $\mathbf{p}$ and $\mathbf{q}$, let $\mathbf{RE}(\mathbf{p} \parallel \mathbf{q})$ denote their relative entropy, i.e. $\mathbf{RE}(\mathbf{p} \parallel \mathbf{q}) = \sum_i p_i \ln(\frac{p_i}{q_i})$.

**Problem Statement.** In a sequence of rounds, for $t = 1, 2, \ldots, T$, we are required to choose a *slate* from a base set $\mathcal{A}$ of $K$ actions. An unordered *slate* is a subset $S \subseteq \mathcal{A}$ of $s$ out of the $K$ actions. An ordered slate is a slate together with an ordering over its $s$ actions; thus, it is a one-to-one mapping $\pi : \{1, 2, \ldots, s\} \rightarrow \mathcal{A}$. Prior to the selection of the slate, the adversary chooses losses[3] for the actions in the slates. Once the slate is chosen, the cost of only the actions in the chosen slate is revealed. This cost is defined in the following manner:

- **Unordered slate.** The adversary chooses a loss vector $\boldsymbol{\ell}(t) \in \mathbb{R}^K$ which specifies a loss $\ell_j(t) \in [-1, 1]$ for every action $j \in \mathcal{A}$. For a chosen slate $S$, only the coordinates $\ell_j(t)$ for $j \in S$ are revealed, and the cost incurred for choosing $S$ is $\sum_{j \in S} \ell_j(t)$.

- **Ordered slate.** The adversary chooses a loss matrix $\mathbf{L}(t) \in \mathbb{R}^{s \times K}$ which specifies a loss $L_{ij}(t) \in [-1, 1]$ for every action $j \in \mathcal{A}$ and every position $i$, $1 \le i \le s$, in the ordering on the slate. For a chosen slate $\pi$, the entries $L_{i, \pi(i)}(t)$ for every position $i$ are revealed, and the cost incurred for choosing $\pi$ is $\sum_{i=1}^s L_{i, \pi(i)}(t)$.

In the unordered slate problem, if slate $S(t)$ is chosen in round $t$, for $t = 1, 2, \ldots, T$, then the regret of the algorithm is defined to be

$$\text{Regret}_T = \sum_{t=1}^T \sum_{j \in S(t)} \ell_j(t) - \min_S \sum_{t=1}^T \sum_{j \in S} \ell_j(t).$$

Here, the subscript $_S$ is used as a shorthand for ranging over all slates $S$. The regret for the ordered slate problem is defined analogously.

Our goal is to design a randomized algorithm for online slate selection such that $\mathbb{E}[\text{Regret}_T] = o(T)$, where the expectation is taken over the internal randomization of the algorithm.

**Competing with policies.** Frequently in applications we have access to $N$ policies which are algorithms that recommend slates to use in every round. These policies might leverage extra information that we have about the losses in the next round. It is therefore beneficial to devise algorithms that have low regret with respect to the best policy in the pool in hindsight, where regret is defined as:

$$\text{Regret}_T = \sum_{t=1}^T \sum_{j \in S(t)} \ell_j(t) - \min_\rho \sum_{t=1}^T \sum_{j \in S_\rho(t)} \ell_j(t).$$

Here, $\rho$ ranges over all policies, $S_\rho(t)$ is the recommendation of policy $\rho$ at time $t$, and $S(t)$ is the algorithm's chosen slate. The regret is defined analogously for ordered slates. More generally, we may allow policies to recommend distributions over slates, and our goal is to minimize the expected regret with respect to the best policy in hindsight, where the expectation is taken over the distribution recommended by the policy as well as the internal randomization of the algorithm.

**Our results.** We are now able to formally state our main results:

**Theorem 2.1.** *There are efficient (running in poly$(s, K)$ time in the no-policies case, and in poly$(s, K, N)$ time with $N$ policies) randomized algorithms achieving the following regret bounds:*

| | Unordered slates | | Ordered slates | |
|---|---|---|---|---|
| *No policies* | $4\sqrt{sK \ln(K/s)T}$ | *(Sec. 3.2)* | $4s\sqrt{K \ln(K)T}$ | *(Sec. 3.3)* |
| *N policies* | $4\sqrt{sK \ln(N)T}$ | *(Sec. 4.1)* | $4s\sqrt{K \ln(N)T}$ | *(Sec. 4.2)* |

To compare, the best bounds obtained for the no-policies case using the more general algorithms [1] and [9] are $O(\sqrt{s^3 K \ln(K/s)T})$ in the unordered slates problem, and $O(s^2 \sqrt{K \ln(K)T})$ in the ordered slates problem. It is also possible, in the no-policies setting, to devise algorithms that have regret bounded by $O(\sqrt{T})$ with high probability, using the upper confidence bounds technique of [3]. We omit these algorithms in this paper for the sake of brevity.

**Algorithm** `MW`$(\mathcal{P})$

**Initialization:** An arbitrary probability distribution $\mathbf{p}(1) \in \mathcal{P}$ on the experts, and some $\eta > 0$.
**For** $t = 1, 2, \ldots, T$:

1. Choose distribution $\mathbf{p}(t)$ over experts, and observe the cost vector $\boldsymbol{\ell}(t)$.
2. Compute the probability vector $\hat{\mathbf{p}}(t+1)$ using the following multiplicative update rule: for every expert $i$,
$$\hat{p}_i(t+1) \;=\; p_i(t)\exp(-\eta\ell_i(t))/Z(t) \tag{1}$$
where $Z(t) = \sum_i p_i(t)\exp(-\eta\ell_i(t))$ is the normalization factor.
3. Set $\mathbf{p}(t+1)$ to be the projection of $\hat{\mathbf{p}}(t+1)$ on the set $\mathcal{P}$ using the **RE** as a distance function, i.e. $\mathbf{p}(t+1) = \arg\min_{\mathbf{p}\in\mathcal{P}} \mathbf{RE}(\mathbf{p} \parallel \hat{\mathbf{p}}(t+1))$.

---

Figure 1: The Multiplicative Weights Algorithm with Restricted Distributions

## 3 Algorithms for the slate problems with no policies

### 3.1 Main algorithmic ideas

Our starting point is the Hedge algorithm for learning online with expert advice. In this setting, on each round $t$, the learner chooses a probability distribution $\mathbf{p}(t)$ over experts, each of which then suffers a (fully observable) loss represented by the vector $\boldsymbol{\ell}(t)$. The learner's loss is then $\mathbf{p}(t) \cdot \boldsymbol{\ell}(t)$.

The main idea of our approach is to apply Hedge (and ideas from bandit variants of it, especially Exp3 [3]) by associating the probability distributions that it selects with mixtures of (ordered or unordered) slates, and thus with the randomized choice of a slate. However, this requires that the selected probability distributions have a particular form, which we describe shortly. We therefore need a special variant of Hedge which uses only distributions $\mathbf{p}(t)$ from some fixed convex subset $\mathcal{P}$ of the simplex of all distributions. The goal then is to minimize regret relative to an arbitrary distribution $\mathbf{p} \in \mathcal{P}$. Such a version of Hedge is given in Figure 1, and a statement of its performance below. This algorithm is implicit in the work of [13, 18].

**Theorem 3.1.** *Assume that $\eta > 0$ is chosen so that for all $t$ and $i$, $\eta\ell_i(t) \geq -1$. Then algorithm* `MW`$(\mathcal{P})$ *generates distributions* $\mathbf{p}(1), \ldots, \mathbf{p}(T) \in \mathcal{P}$, *such that for any* $\mathbf{p} \in \mathcal{P}$,

$$\sum_{t=1}^{T} \boldsymbol{\ell}(t) \cdot \mathbf{p}(t) - \boldsymbol{\ell}(t) \cdot \mathbf{p} \;\leq\; \eta \sum_{t=1}^{T} (\boldsymbol{\ell}(t))^{\mathbf{2}} \cdot \mathbf{p}(t) + \frac{\mathbf{RE}(\mathbf{p} \parallel \mathbf{p}(1))}{\eta}.$$

*Here, $(\boldsymbol{\ell}(t))^{\mathbf{2}}$ is the vector that is the coordinate-wise square of $\boldsymbol{\ell}(t)$.*

### 3.2 Unordered slates with no policies

To apply the approach described above, we need a way to compactly represent the set of distributions over slates. We do this by embedding slates as points in some high-dimensional Euclidean space, and then giving a compact representation of the convex hull of the embedded points. Specifically, we represent an unordered slate $S$ by its indicator vector $\mathbf{1}_S \in \mathbb{R}^K$, which is 1 for all coordinates $j \in S$, and 0 for all others. The convex hull $\mathcal{X}$ of all such $\mathbf{1}_S$ vectors can be succinctly described [18] as the convex polytope defined by the linear constraints $\sum_{j=1}^{K} x_j = s$ and $x_j \geq 0$ for $j = 1, \ldots, K$. An algorithm is given in [18] (Algorithm 2) to decompose any vector $\mathbf{x} \in \mathcal{X}$ into a convex combination of at most $K$ indicator vectors $\mathbf{1}_S$. We embed the convex hull $\mathcal{X}$ of all the $\mathbf{1}_S$ vectors in the simplex of distributions over the $K$ actions simply by scaling down all coordinates by $s$ so that they sum to 1. Let $\mathcal{P}$ be this scaled down version of $\mathcal{X}$. Our algorithm is given in Figure 2.

Step 3 of `MW`$(\mathcal{P})$ requires us to compute the $\arg\min_{\mathbf{p}\in\mathcal{P}} \mathbf{RE}(\mathbf{p} \parallel \hat{\mathbf{p}}(t+1))$, which can be solved by convex programming. A linear time algorithm is given in [13], and a simple algorithm (from [18]) is the following: find the least index $k$ such that clipping the largest $k$ coordinates of $\mathbf{p}$ to $\frac{1}{s}$ and rescaling the rest of the coordinates to sum up to $1 - \frac{k}{s}$ ensures that all coordinates are at most $\frac{1}{s}$, and output the probability vector thus obtained. This can be implemented by sorting the coordinates, and so it takes $O(K \log(K))$ time.

---

**Bandit Algorithm for Unordered Slates**

**Initialization:** Start an instance of $\mathtt{MW}(\mathcal{P})$ with the uniform initial distribution $\mathbf{p}(1) = \frac{1}{K}\mathbf{1}$. Set $\eta = \sqrt{\frac{(1-\gamma)s\ln(K/s)}{KT}}$, and $\gamma = \sqrt{\frac{(K/s)\ln(K/s)}{T}}$.

**For** $t = 1, 2, \ldots, T$:

1. Obtain the distribution $\mathbf{p}(t)$ from $\mathtt{MW}(\mathcal{P})$.
2. Set $\mathbf{p}'(t) = (1-\gamma)\mathbf{p}(t) + \frac{\gamma}{K}\mathbf{1}_{\mathcal{A}}$.
3. Note that $\mathbf{p}'(t) \in \mathcal{P}$. Decompose $s\mathbf{p}'(t)$ as a convex combination of slate vectors $\mathbf{1}_S$ corresponding to slates $S$ as $s\mathbf{p}'(t) = \sum_S q_S \mathbf{1}_S$, where $q_S > 0$ and $\sum_S q_S = 1$.
4. Choose a slate $S$ to display with probability $q_S$, and obtain the loss $\ell_j(t)$ for all $j \in S$.
5. Set $\hat{\ell}_j(t) = \ell_j(t)/(sp'_j(t))$ if $j \in S$, and 0 otherwise.
6. Send $\hat{\boldsymbol{\ell}}(t)$ as the loss vector to $\mathtt{MW}(\mathcal{P})$.

---

Figure 2: The Bandit Algorithm with Unordered Slates

We now prove the regret bound of Theorem 2.1. We use the notation $\mathbb{E}_t[X]$ to denote the expectation of a random variable $X$ conditioned on all the randomness chosen by the algorithm up to round $t$, assuming that $X$ is measurable with respect to this randomness. We note the following facts: $\mathbb{E}_t[\hat{\ell}_j(t)] = \sum_{S \ni j} q_S \cdot \frac{\ell_j(t)}{sp'_j(t)} = \ell_j(t)$, since $p'_j(t) = \sum_{S \ni j} q_S \cdot \frac{1}{s}$. This immediately implies that $\mathbb{E}_t[\hat{\boldsymbol{\ell}}(t) \cdot \mathbf{p}(t)] = \boldsymbol{\ell}(t) \cdot \mathbf{p}(t)$ and $\mathbb{E}[\hat{\boldsymbol{\ell}}(t) \cdot \mathbf{p}] = \boldsymbol{\ell}(t) \cdot \mathbf{p}$, for any fixed distribution $\mathbf{p}$.

Note that if we decompose a distribution $\mathbf{p} \in \mathcal{P}$ as a convex combination of $\frac{1}{s}\mathbf{1}_S$ vectors and randomly choose a slate $S$ according to its weight in the combination, then the expected loss, averaged over the $s$ actions chosen, is $\boldsymbol{\ell}(t) \cdot \mathbf{p}$. We can bound the difference between the expected loss (averaged over the $s$ actions) in round $t$ suffered by the algorithm, $\boldsymbol{\ell}(t) \cdot \mathbf{p}'(t)$, and $\boldsymbol{\ell}(t) \cdot \mathbf{p}(t)$ as follows:

$$\boldsymbol{\ell}(t) \cdot \mathbf{p}'(t) - \boldsymbol{\ell}(t) \cdot \mathbf{p}(t) = \sum_j \ell_j(t)(p'_j(t) - p_j(t)) \leq \sum_j \ell_j(t) \cdot \frac{\gamma}{K} \leq \gamma.$$

Using this bound and Theorem 3.1, if $S^\star = \arg\min_S \sum_t \boldsymbol{\ell}(t) \cdot \frac{1}{s}\mathbf{1}_S$, we have

$$\frac{\mathbb{E}[\text{Regret}_T]}{s} = \sum_t \boldsymbol{\ell}(t) \cdot \mathbf{p}'(t) - \boldsymbol{\ell}(t) \cdot \frac{1}{s}\mathbf{1}_{S^\star} \leq \eta \sum_t \mathbb{E}[(\hat{\boldsymbol{\ell}}(t))^2 \cdot \mathbf{p}(t)] + \frac{\mathbf{RE}(\frac{1}{s}\mathbf{1}_{S^\star} \parallel \mathbf{p}(1))}{\eta} + \gamma T.$$

We note that the leading factor of $\frac{1}{s}$ on the expected regret is due to the averaging over the $s$ positions. We now bound the terms on the RHS. First, we have

$$
\begin{aligned}
\mathbb{E}_t[(\hat{\boldsymbol{\ell}}(t))^2 \cdot \mathbf{p}(t)] &= \sum_S q_S \cdot \left[ \sum_{j \in S} \frac{(\ell_j(t))^2 p_j(t)}{(sp'_j(t))^2} \right] \\
&= \sum_j \left[ \frac{(\ell_j(t))^2 p_j(t)}{(sp'_j(t))^2} \right] \cdot \sum_{S \ni j} q_S = \sum_j \left[ \frac{(\ell_j(t))^2 p_j(t)}{(sp'_j(t))^2} \right] \cdot sp'_j(t) \leq \frac{K}{s(1-\gamma)},
\end{aligned}
$$

because $\frac{p_j(t)}{p'_j(t)} \leq \frac{1}{1-\gamma}$, and all $|\ell_j(t)| \leq 1$.

$$\mathbb{E}[\text{Regret}_T] \leq \eta\frac{KT}{1-\gamma} + \frac{s\ln(K/s)}{\eta} + s\gamma T \leq 4\sqrt{sK\ln(K/s)T},$$

by setting $\eta = \sqrt{\frac{(1-\gamma)s\ln(K/s)}{KT}}$ and $\gamma = \sqrt{\frac{(K/s)\ln(K/s)}{T}}$.

It remains to verify that $\eta\hat{\ell}_j(t) \geq -1$ for all $i$ and $t$. We know that $\hat{\ell}_j(t) \geq -\frac{K}{s\gamma}$, because $p'_j(t) \geq \frac{\gamma}{K}$, so all we need to check is that $\sqrt{\frac{(1-\gamma)s\ln(K/s)}{KT}} \leq \frac{s\gamma}{K}$, which is true for our choice of $\gamma$.

---

**Bandit Algorithm for Ordered Slates**

**Initialization:** Start an instance of $\texttt{MW}(\mathcal{P})$ with the uniform initial distribution $\mathbf{p}(1) = \frac{1}{sK}\mathbf{1}$. Set $\eta = \sqrt{\frac{(1-\gamma)\ln(K)}{KT}}$ and $\gamma = \sqrt{\frac{K\ln(K)}{T}}$. **For** $t = 1, 2, \dots, T$:

1. Obtain the distribution $\mathbf{p}(t)$ from $\texttt{MW}(\mathcal{P})$.
2. Set $\mathbf{p}'(t) = (1 - \gamma)\mathbf{p}(t) + \frac{\gamma}{sK}\mathbf{1}_{\mathcal{A}}$.
3. Note that $\mathbf{p}'(t) \in \mathcal{P}$, and so $s\mathbf{p}'(t) \in \mathcal{M}$. Decompose $s\mathbf{p}'(t)$ as a convex combination of $\mathbf{M}^{\pi}$ matrices corresponding to ordered slates $\pi$ as $s\mathbf{p}'(t) = \sum_{\pi} q_{\pi}\mathbf{M}_{\pi}$, where $q_{\pi} > 0$ and $\sum_{\pi} q_{\pi} = 1$.
4. Choose a slate $\pi$ to display w.p. $q_{\pi}$, and obtain the loss $L_{i,\pi(i)}(t)$ for all $1 \le i \le s$.
5. Construct the loss matrix $\hat{\mathbf{L}}(t)$ as follows: for $1 \le i \le s$, set $\hat{L}_{i,\pi(i)}(t) = \frac{L_{i,\pi(i)}(t)}{sp'_{i,\pi(i)}(t)}$, and all other entries are 0.
6. Send $\hat{\mathbf{L}}(t)$ as the loss vector to $\texttt{MW}(\mathcal{P})$.

---

Figure 3: Bandit Algorithm for Ordered Slates

## 3.3 Ordered slates with no policies

A similar approach can be used for ordered slates. Here, we represent an ordered slate $\pi$ by the *subpermutation matrix* $\mathbf{M}^{\pi} \in \mathbb{R}^{s \times K}$ which is defined as follows: for $i = 1, 2, \dots, s$, we have $M_{i,\pi(i)}^{\pi} = 1$, and all other entries are 0. In [7, 16], it is shown that the convex hull $\mathcal{M}$ of all the $\mathbf{M}^{\pi}$ matrices is the convex polytope defined by the linear constraints: $\sum_{j=1}^{K} M_{ij} = 1$ for $i = 1, \dots, s$; $\sum_{i=1}^{s} M_{ij} \le 1$ for $j = 1, \dots, K$; and $M_{ij} \ge 0$ for $i = 1, \dots, s$ and $j = 1, \dots, K$. Clearly, all subpermutation matrices $\mathbf{M}^{\pi} \in \mathcal{M}$. To complete the characterization of the convex hull, we can show (details omitted) that given any matrix $\mathbf{M} \in \mathcal{M}$, we can efficiently decompose it into a convex combination of at most $K^2$ subpermutation matrices.

We identify matrices in $\mathbb{R}^{s \times K}$ with vectors in $\mathbb{R}^{sK}$ in the obvious way. We embed $\mathcal{M}$ in the simplex of distributions in $\mathbb{R}^{sK}$ simply by scaling all the entries down by $s$ so that their sum equals one. Let $\mathcal{P}$ be this scaled down version of $\mathcal{M}$. Our algorithm is given in Figure 3.

The projection in step 3 of $\texttt{MW}(\mathcal{P})$ can be computed simply by solving the convex program. In practice, however, noticing that the relative entropy projection is a Bregman projection, the cyclic projections method of Bregman [6, 8] is likely to work faster. Adapted to the specific problem at hand, this method works as follows (see [8] for details): first, for every column $j$, initialize a dual variable $\lambda_j = 1$. Then, alternate between row phases and column phases. In a row phase, iterate over all rows, and rescale them to make them sum to $\frac{1}{s}$. The column phase is a little more complicated: first, for every column $j$, compute the scaling factor $\alpha$ to make it sum to $\frac{1}{s}$. Set $\alpha' = \min\{\lambda_j, \alpha\}$, and scale the column by $\alpha'$, and update $\lambda_j \leftarrow \lambda_j/\alpha'$. Repeat these alternating row and column phases until convergence to within the desired tolerance.

The regret bound analysis is similar to that of Section 3.2. We have $\mathbb{E}_t[\hat{L}_{ij}(t)] = \sum_{\pi:\pi(i)=j} q_{\pi} \cdot \frac{L_{ij}(t)}{sp'_{ij}} = L_{ij}(t)$, and hence $\mathbb{E}_t[\hat{\mathbf{L}}(t) \bullet \mathbf{p}(t)] = \mathbf{L}(t) \bullet \mathbf{p}(t)$ and $\mathbb{E}[\hat{\mathbf{L}}(t) \bullet \mathbf{p}] = \mathbf{L}(t) \bullet \mathbf{p}$. We can show also that $\mathbf{L}(t) \bullet \mathbf{p}'(t) - \mathbf{L}(t) \bullet \mathbf{p}(t) \le \gamma$.

Using this bound and Theorem 3.1, if $\pi^{\star} = \arg\min_{\pi} \sum_t \mathbf{L}(t) \bullet \frac{1}{s}\mathbf{M}^{\pi}$, we have

$$\frac{\mathbb{E}[\text{Regret}_T]}{s} = \sum_t \mathbf{L}(t)\bullet\mathbf{p}'(t) - \mathbf{L}(t)\bullet\frac{1}{s}\mathbf{M}^{\pi^{\star}} \le \eta\sum_t \mathbb{E}[(\hat{\mathbf{L}}(t))^2\bullet\mathbf{p}(t)] + \frac{\mathbf{RE}(\frac{1}{s}\mathbf{M}^{\pi^{\star}}\|\mathbf{p}(1))}{\eta} + \gamma T.$$

We now bound the terms on the RHS. First, we have

$$\mathbb{E}_t[(\hat{\mathbf{L}}(t))^2 \bullet \mathbf{p}(t)] = \sum_{\pi} q_{\pi} \cdot \left[\sum_{i=1}^{s} \frac{(L_{i,\pi(i)}(t))^2 p_{i,\pi(i)}(t)}{(sp'_{i,\pi(i)}(t))^2}\right] = \sum_{i=1}^{s}\sum_{j=1}^{K} \left[\frac{(L_{ij}(t))^2 p_{ij}(t)}{(sp'_{ij}(t))^2}\right] \cdot \sum_{\pi:\pi(i)=j} q_{\pi}$$

---

**Bandit Algorithm for Unordered Slates With Policies**

**Initialization:** Start an instance of `MW` with no restrictions over the set of distributions over the $N$ policies, with the initial distribution $\mathbf{r}(1) = \frac{1}{N}\mathbf{1}$. Set $\eta = \sqrt{\frac{(1-\gamma)s\ln(N)}{KT}}$, and $\gamma = \sqrt{\frac{(K/s)\ln(N)}{T}}$.

**For** $t = 1, 2, \ldots, T$:

1. Obtain the distribution over policies $\mathbf{r}(t)$ from `MW`, and the recommended distribution over slates $\boldsymbol{\phi}_\rho(t) \in \mathcal{P}$ for each policy $\rho$.

2. Compute the distribution $\mathbf{p}(t) = \sum_{\rho=1}^{N} r_\rho(t)\boldsymbol{\phi}_\rho(t)$.

3. Set $\mathbf{p}'(t) = (1-\gamma)\mathbf{p}(t) + \frac{\gamma}{K}\mathbf{1}$.

4. Note that $\mathbf{p}'(t) \in \mathcal{P}$. Decompose $s\mathbf{p}'(t)$ as a convex combination of slate vectors $\mathbf{1}_S$ corresponding to slates $S$ as $s\mathbf{p}'(t) = \sum_S q_S \mathbf{1}_S$, where $q_S > 0$ and $\sum_S q_S = 1$.

5. Choose a slate $S$ to display with probability $q_S$, and obtain the loss $\ell_j(t)$ for all $j \in S$.

6. Set $\hat{\ell}_j(t) = \ell_j(t)/sp'_j(t)$ if $j \in S$, and 0 otherwise.

7. Set the loss of policy $\rho$ to be $\lambda_\rho(t) = \hat{\boldsymbol{\ell}}(t) \cdot \boldsymbol{\phi}_\rho(t)$ in the MW algorithm.

---

Figure 4: Bandit Algorithm for Unordered Slates With Policies

$$= \sum_{i=1}^{s}\sum_{j=1}^{K}\left[\frac{(L_{ij}(t))^2 p_{ij}(t)}{(sp'_{ij}(t))^2}\right]\cdot sp'_{ij}(t) \le \frac{K}{1-\gamma},$$

because $\frac{p_{ij}(t)}{p'_{ij}(t)} \le \frac{1}{1-\gamma}$, all $|L_{ij}(t)| \le 1$.

Finally, we have $\mathbf{RE}(\frac{1}{s}\mathbf{M}^{\pi^\star} \| \mathbf{p}(1)) = \ln(K)$. Plugging these bounds into the bound of Theorem 3.1, we get the stated regret bound from Theorem 2.1:

$$\mathbb{E}[\text{Regret}_T] \le \eta\frac{sKT}{1-\gamma} + \frac{s\ln(K)}{\eta} + s\gamma T \le 4s\sqrt{K\ln(K)T},$$

by setting $\eta = \sqrt{\frac{(1-\gamma)\ln(K)}{KT}}$ and $\gamma = \sqrt{\frac{K\ln(K)}{T}}$, which satisfy the necessary technical conditions.

## 4 Competing with a set of policies

### 4.1 Unordered Slates with $N$ Policies

In each round, every policy $\rho$ recommends a distribution over slates $\boldsymbol{\phi}_\rho(t) \in \mathcal{P}$, where $\mathcal{P}$ is the $\mathcal{X}$ scaled down by $s$ as in Section 3.2. Our algorithm is given in Figure 4.

Again the regret bound analysis is along the lines of Section 3.2. We have for any $j$, $\mathbb{E}_t[\hat{\ell}_j(t)] = \sum_{S\ni j} q_S \cdot \frac{\ell_j(t)}{sp'_j(t)} = \ell_j(t)$. Thus, $\mathbb{E}_t[\lambda_\rho(t)] = \boldsymbol{\ell}(t)\cdot\boldsymbol{\phi}_\rho(t)$, and hence $\mathbb{E}_t[\lambda(t)\cdot\mathbf{r}(t)] = \sum_\rho(\boldsymbol{\ell}(t)\cdot\boldsymbol{\phi}_\rho(t))r_\rho(t) = \boldsymbol{\ell}(t)\cdot\mathbf{p}(t)$. We can also show as before that $\boldsymbol{\ell}(t)\cdot\mathbf{p}'(t) - \boldsymbol{\ell}(t)\cdot\mathbf{p}(t) \le \gamma$.

Using this bound and Theorem 3.1, if $\rho^\star = \arg\min_\rho \sum_t \boldsymbol{\ell}(t)\cdot\boldsymbol{\phi}_\rho(t)$, we have

$$\frac{\mathbb{E}[\text{Regret}_T]}{s} = \sum_t \boldsymbol{\ell}(t)\cdot\mathbf{p}'(t) - \boldsymbol{\ell}(t)\cdot\boldsymbol{\phi}_{\rho^\star}(t) \le \eta\sum_t \mathbb{E}[(\lambda(t))^2\cdot\mathbf{r}(t)] + \frac{\mathbf{RE}(\mathbf{e}_{\rho^\star}\|\mathbf{r}(1))}{\eta} + \gamma T,$$

where $\mathbf{e}_{\rho^\star}$ is the distribution (vector) that is concentrated entirely on policy $\rho^\star$.

We now bound the terms on the RHS. First, we have

$$\mathbb{E}_t[(\lambda(t))^2\cdot\mathbf{r}(t)] = \mathbb{E}_t\left[\sum_\rho \lambda_\rho(t)^2 r_\rho(t)\right] = \mathbb{E}_t\left[\sum_\rho (\hat{\boldsymbol{\ell}}(t)\cdot\boldsymbol{\phi}_\rho(t))^2 r_\rho(t)\right]$$

$$\le \mathbb{E}_t\left[\sum_\rho ((\hat{\boldsymbol{\ell}}(t))^2\cdot\boldsymbol{\phi}_\rho(t))r_\rho(t)\right] = \mathbb{E}_t[(\hat{\boldsymbol{\ell}}(t))^2\cdot\mathbf{p}(t)] \le \frac{K}{s(1-\gamma)}.$$

---

**Bandit Algorithm for Ordered Slates with Policies**

**Initialization:** Start an instance of MW with no restrictions, over the set of distributions over the $N$ policies, starting with $\mathbf{r}(1) = \frac{1}{N}\mathbf{1}$. Set $\eta = \sqrt{\frac{(1-\gamma)\ln(N)}{KT}}$ and $\gamma = \sqrt{\frac{K\ln(N)}{T}}$.

**For** $t = 1, 2, \ldots, T$:

1. Obtain the distribution over policies $\mathbf{r}(t)$ from MW, and the recommended distribution over ordered slates $\phi_\rho(t) \in \mathcal{P}$ for each policy $\rho$.

2. Compute the distribution $\mathbf{p}(t) = \sum_{\rho=1}^{N} r_\rho(t)\phi_\rho(t)$.

3. Set $\mathbf{p}'(t) = (1-\gamma)\mathbf{p}(t) + \frac{\gamma}{sK}\mathbf{1}_\mathcal{A}$.

4. Note that $\mathbf{p}'(t) \in \mathcal{P}$, and so $s\mathbf{p}'(t) \in \mathcal{X}$. Decompose $s\mathbf{p}'(t)$ as a convex combination of $\mathbf{M}^\pi$ matrices corresponding to ordered slates $\pi$ as $s\mathbf{p}'(t) = \sum_\pi q_\pi \mathbf{M}_\pi$, where $q_\pi > 0$ and $\sum_\pi q_\pi = 1$.

5. Choose a slate $\pi$ to display w.p. $q_\pi$, and obtain the loss $L_{i,\pi(i)}(t)$ for all $1 \le i \le s$.

6. Construct the loss matrix $\hat{\mathbf{L}}(t)$ as follows: for $1 \le i \le s$, set $\hat{L}_{i,\pi(i)}(t) = \frac{L_{i,\pi(i)}(t)}{sp'_{i,\pi(i)}(t)}$, and all other entries are 0.

7. Set the loss of policy $\rho$ to be $\lambda_\rho(t) = \hat{\mathbf{L}}(t) \bullet \phi_\rho(t)$ in the MW algorithm.

---

Figure 5: Bandit Algorithm for Ordered Slates with Policies

The first inequality above follows from Jensen's inequality, and the second one is proved exactly as in Section 3.2. Finally, we have $\mathbf{RE}(\mathbf{e}_{\rho^\star} \parallel \mathbf{p}(1)) = \ln(N)$. Plugging these bounds into the bound above, we get the stated regret bound from Theorem 2.1:

$$\mathbb{E}[\text{Regret}_T] \le \eta \frac{KT}{1-\gamma} + \frac{s\ln(N)}{\eta} + s\gamma T \le 4\sqrt{sK\ln(N)T},$$

by setting $\eta = \sqrt{\frac{(1-\gamma)s\ln(N)}{KT}}$ and $\gamma = \sqrt{\frac{(K/s)\ln(N)}{T}}$, which satisfy the necessary technical conditions.

## 4.2 Ordered Slates with $N$ Policies

In each round, every policy $\rho$ recommends a distribution over ordered slates $\phi_\rho(t) \in \mathcal{P}$, where $\mathcal{P}$ is $\mathcal{M}$ scaled down by $s$ as in Section 3.3. Our algorithm is given in Figure 5.

The regret bound analysis is exactly along the lines of that in Section 4.1, with $\mathbf{L}(t)$ and $\hat{\mathbf{L}}(t)$ playing the roles of $\ell(t)$ and $\hat{\ell}(t)$ respectively, with the inequalities from Section 3.3. We omit the details for brevity. We get the stated regret bound from Theorem 2.1:

$$\mathbb{E}[\text{Regret}_T] \le 4s\sqrt{K\ln(N)T}.$$

## 5 Conclusions and Future Work

In this paper, we presented efficient algorithms for the unordered and ordered slate problems with regret bounds of $O(\sqrt{T})$, in the presence and and absence of policies, employing the technique of Bregman projections on a convex set representing the convex hull of slate vectors.

Possible future work on this problem is in two directions. The first direction is to handle other user models for the loss matrices, such as models incorporating the following sort of interaction between the chosen actions: if two very similar ads are shown, and the user clicks on one, then the user is less likely to click on the other. Our current model essentially assumes no interaction.

The second direction is to derive high probability $O(\sqrt{T})$ regret bounds for the slate problems in the presence of policies. The techniques of [3] only give such algorithms in the no-policies setting.

## Footnotes

[1]The unordered slate problem is a special case of the ordered slate problem for which all positional factors are equal. However, the bound on the regret that we get when we consider the unordered slate problem separately is a factor of $\tilde{O}(\sqrt{s})$ better than when we treat it as a special case of the ordered slate problem.

[2]The difference in the regret bounds can be attributed to the definition of regret in the stochastic and non-stochastic settings. In the stochastic setting, we compare the algorithm's expected reward to that of the arm with the largest expected reward, with the expectation taken over the reward distribution.

[3] Note that we switch to losses rather than rewards to be consistent with most recent literature on online learning. Since we allow negative losses, we can easily deal with rewards as well.

## References

[1] ABERNETHY, J., HAZAN, E., AND RAKHLIN, A. Competing in the dark: An efficient algorithm for bandit linear optimization. In *COLT* (2008), pp. 263–274.

[2] AUER, P., CESA-BIANCHI, N., AND FISCHER, P. Finite-time analysis of the multiarmed bandit problem. *Machine Learning 47*, 2-3 (2002), 235–256.

[3] AUER, P., CESA-BIANCHI, N., FREUND, Y., AND SCHAPIRE, R. E. The nonstochastic multiarmed bandit problem. *SIAM J. Comput. 32*, 1 (2002), 48–77.

[4] AWERBUCH, B., AND KLEINBERG, R. Online linear optimization and adaptive routing. *J. Comput. Syst. Sci. 74*, 1 (2008), 97–114.

[5] BARTLETT, P. L., DANI, V., HAYES, T. P., KAKADE, S., RAKHLIN, A., AND TEWARI, A. High-probability regret bounds for bandit online linear optimization. In *COLT* (2008), pp. 335–342.

[6] BREGMAN, L. The relaxation method of finding the common point of convex sets and its application to the solution of problems in convex programming. *USSR Comp. Mathematics and Mathematical Physics 7* (1967), 200–217.

[7] BRUALDI, R. A., AND LEE, G. M. On the truncated assignment polytope. *Linear Algebra and its Applications 19* (1978), 33–62.

[8] CENSOR, Y., AND ZENIOS, S. *Parallel optimization*. Oxford University Press, 1997.

[9] CESA-BIANCHI, N., AND LUGOSI, G. Combinatorial bandits. In *COLT* (2009).

[10] GYÖRGY, A., LINDER, T., LUGOSI, G., AND OTTUCSÁK, G. The on-line shortest path problem under partial monitoring. *Journal of Machine Learning Research 8* (2007), 2369–2403.

[11] HAZAN, E., AND KALE, S. Better algorithms for benign bandits. In *SODA* (2009), pp. 38–47.

[12] HELMBOLD, D. P., AND WARMUTH, M. K. Learning permutations with exponential weights. In *COLT* (2007), pp. 469–483.

[13] HERBSTER, M., AND WARMUTH, M. K. Tracking the best linear predictor. *Journal of Machine Learning Research 1* (2001), 281–309.

[14] KOOLEN, W. M., WARMUTH, M. K., AND KIVINEN, J. Hedging structured concepts. In *COLT* (2010).

[15] LAI, T., AND ROBBINS, H. Asymptotically efficient adaptive allocation rules. *Advances in Applied Mathematics 6* (1985), 4–22.

[16] MENDELSOHN, N. S., AND DULMAGE, A. L. The convex hull of sub-permutation matrices. *Proceedings of the American Mathematical Society 9*, 2 (Apr 1958), 253–254.

[17] UCHIYA, T., NAKAMURA, A., AND KUDO, M. Algorithms for adversarial bandit problems with multiple plays. In *ALT* (2010), pp. 375–389.

[18] WARMUTH, M. K., AND KUZMIN, D. Randomized PCA algorithms with regret bounds that are logarithmic in the dimension. In *In Proc. of NIPS* (2006).

